# Combining Neural Network Regression Estimates with Regularized Linear Weights

Christopher J. Merz and Michael J. Pazzani

Dept. of Information and Computer Science
University of California, Irvine, CA 92717-3425 U.S.A.
{cmerz,pazzani}@ics.uci.edu

Category: Algorithms and Architectures.

## Abstract

When combining a set of learned models to form an improved estimator, the issue of redundancy or multicollinearity in the set of models must be addressed. A progression of existing approaches and their limitations with respect to the redundancy is discussed. A new approach, PCR*, based on principal components regression is proposed to address these limitations. An evaluation of the new approach on a collection of domains reveals that: 1) PCR* was the most robust combination method as the redundancy of the learned models increased, 2) redundancy could be handled without eliminating any of the learned models, and 3) the principal components of the learned models provided a continuum of "regularized" weights from which PCR* could choose.

## 1 INTRODUCTION

Combining a set of learned models[1] to improve classification and regression estimates has been an area of much research in machine learning and neural networks [Wolpert, 1992, Merz, 1995, Perrone and Cooper, 1992, Leblanc and Tibshirani, 1993, Breiman, 1992, Meir, 1995, Krogh and Vedelsby, 1995, Tresp, 1995, Chan and Stolfo, 1995]. The challenge of this problem is to decide which models to rely on for prediction and how much weight to give each.

The goal of combining learned models is to obtain a more accurate prediction than can be obtained from any single source alone. One major issue in combining a set of learned models is redundancy. *Redundancy* refers to the amount of agreement or linear dependence between models when making a set of predictions. The more the set agrees, the more redundancy is present. In statistical terms, this is referred to as the multicollinearity problem.

The focus of this paper is to explore and evaluate the properties of existing methods for combining regression estimates (Section 2), and to motivate the need for more advanced methods which deal with multicollinearity in the set of learned models (Section 3). In particular, a method based on principal components regression (PCR, [Draper and Smith, 1981]) is described, and is evaluated emperically demonstrating the it is a robust and efficient method for finding a set of combining weights with low prediction error (Section 4). Finally, Section 5 draws some conclusions.

## 2  MOTIVATION

The problem of combining a set of learned models is defined using the terminology of [Perrone and Cooper, 1992]. Suppose two sets of data are given: a training set $\mathcal{D}_{Train} = (x_m, y_m)$ and a test set $\mathcal{D}_{Test} = (x_l, y_l)$. Now suppose $\mathcal{D}_{Train}$ is used to build a set of functions, $\mathcal{F} = f_i(x)$, each element of which approximates $f(x)$. The goal is to find the best approximation of $f(x)$ using $\mathcal{F}$.

To date, most approaches to this problem limit the space of approximations of $f(x)$ to linear combinations of the elements of $\mathcal{F}$, i.e.,

$$\hat{f}(x) = \sum_{i=1}^{N} \alpha_i f_i(x)$$

where $\alpha_i$ is the coefficient or weight of $f_i(x)$.

The focus of this paper is to evaluate and address the limitations of these approaches. To do so, a brief summary of these approaches is now provided progressing from simpler to more complex methods pointing out their limitations along the way.

The simplest method for combining the members of $\mathcal{F}$ is by taking the unweighted average, (i.e., $\alpha_i = 1/N$). Perrone and Cooper refer to this as the Basic Ensemble Method (BEM), written as

$$f_{BEM} = 1/N \sum_{i=1}^{N} f_i(x)$$

This equation can also be written in terms of the *misfit function* for each $f_i(x)$. These functions describe the deviations of the elements of $\mathcal{F}$ from the true solution and are written as

$$m_i(x) = f(x) - f_i(x).$$

Thus,

$$f_{BEM} = f(x) - 1/N \sum_{i=1}^{N} m_i(x).$$

Perrone and Cooper show that as long as the $m_i(x)$ are mutually independent with zero mean, the error in estimating $f(x)$ can be made arbitrarily small by increasing the population size of $\mathcal{F}$. Since these assumptions break down in practice,

they developed a more general approach which finds the "optimal"[2] weights while allowing the $m_i(x)$'s to be correlated and have non-zero means. This Generalized Ensemble Method (GEM) is written as

$$f_{GEM} = \sum_{i=1}^{N} \alpha_i f_i(x) = f(x) - \sum_{i=1}^{N} \alpha_i m_i(x)$$

where

$$\alpha_i = \frac{\sum_{j=1}^{N} C_{ij}^{-1}}{\sum_{k=1}^{N} \sum_{j=1}^{N} C_{kj}^{-1}} \text{ and } C_{ij} = E[m_i(x) m_j(x)].$$

$C$ is the symmetric sample covariance matrix for the misfit function and the goal is to minimize $\sum_{i,j}^{N} \alpha_i \alpha_j C_{ij}$. Note that the misfit functions are calculated on the training data and $f(x)$ is not required. The main disadvantage to this approach is that it involves taking the inverse of $C$ which can be "unstable". That is, redundancy in the members of $\mathcal{F}$ leads to linear dependence in the rows and columns of $C$ which in turn leads to unreliable estimates of $C^{-1}$.

To circumvent this sensitivity redundancy, Perrone and Cooper propose a method for discarding member(s) of $\mathcal{F}$ when the strength of its agreement with another member exceeds a certain threshold. Unfortunately, this approach only checks for linear dependence (or redundancy) between pairs of $f_i(x)$ and two $f_i(x)$ for $i \neq j$. In fact, $f_i(x)$ could be a linear combination of several other members of $\mathcal{F}$ and the instability problem would be manifest. Also, depending on how high the threshold is set, a member of $\mathcal{F}$ could be discarded while still having some degree of uniqueness and utility. An ideal method for weighting the members of $\mathcal{F}$ would neither discard any models nor suffer when there is redundancy in the model set.

The next approach reviewed is linear regression (LR)[3] which also finds the "optimal" weights for the $f_i(x)$ with respect to the training data. In fact, GEM and LR are both considered "optimal" because they are closely related in that GEM is a form of linear regression with the added constraint that $\sum_{i=1}^{N} \alpha_i = 1$. The weights for LR are found as follows[4],

$$f_{LR} = \sum_{i=1}^{N} \alpha_i f_i(x)$$

where

$$\alpha = (f^T f)^{-1} f^T F \quad f_{ji} = f_i(x_j) 1 \leq j \leq M \text{ and } F_j = f(x_j).$$

Like GEM, LR and LRC are subject to the multicollinearity problem because finding the $\alpha_i$'s involves taking the inverse of a matrix. That is, if the $f$ matrix is composed of $f_i(x)$ which strongly agree with other members of $\mathcal{F}$, some linear dependence will be present.

Given the limitations of these methods, the goal of this research was to find a method which finds weights for the learned models with low prediction error without discarding any of the original models, and without being subject to the multicollinearity problem.

# 3   METHODS FOR HANDLING MULTICOLLINEARITY

In the abovementioned methods, multicollinearity leads to inflation of the variance of the estimated weights, $\alpha$. Consequently, the weights obtained from fitting the model to a particular sample may be far from their true values. To circumvent this problem, approaches have been developed which: 1) constrain the estimated regression coefficients so as to improve prediction performance (i.e., ridge regression, RIDGE [Montgomery and Friedman 1993], and principal components regression), 2) search for the coefficients via gradient descent procedures (i.e., Widrow-Hoff learning, GD and EG+- [Kivinen and Warmuth, 1994]), or build models which make decorrelated errors by adjusting the bias of the learning algorithm [Opitz and Shavlik, 1995] or the data which it sees [Meir, 1995]. The third approach ameliorates, but does not solve, the problem because redundancy is an inherent part of the task of combining estimators.

The focus of this paper is on the first approach. Leblanc and Tibshirani [Leblanc and Tibshirani, 1993] have proposed several ways of constraining or *regularizing* the weights to help produce estimators with lower prediction error:

1. Shrink $\hat{\alpha}$ towards $(1/K, 1/K, \ldots, 1/K)^T$ where $K$ is the number of learned models.
2. $\sum_{i=1}^{N} \alpha_i = 1$
3. $\alpha_i \geq 0, i = 1, 2 \ldots K$

Breiman [Breiman, 1992] provides an intuitive justification for these constraints by pointing out that the more strongly they are satisfied, the more interpolative the weighting scheme is. In the extreme case, a uniformly weighted set of learned models is likely to produce a prediction *between* the maximum and minimum predicted values of the learned models. Without these constraints, there is no guarantee that the resulting predictor will stay near that range and generalization may be poor. The next subsection describes a variant of principal components regression and explains how it provides a continuum of regularized weights for the original learned models.

## 3.1   PRINCIPAL COMPONENTS REGRESSION

When dealing with the above mentioned multicollinearity problem, principal components regression [Draper and Smith, 1981] may be used to summarize and extract the "relevant" information from the learned models. The main idea of PCR is to map the original learned models to a set of (independent) principal components in which each component is a linear combination of the original learned models, and then to build a regression equation using the best subset of the principal components to predict $f(x)$.

The advantage of this representation is that the components are sorted according to how much information (or variance) from the original learned models for which they account. Given this representation, the goal is to choose the number of principal components to include in the final regression by retaining the first $k$ which meet a preselected stopping criteria. The basic approach is summarized as follows:

1. Do a principal components analysis (PCA) on the covariance matrix of the learned models' predictions on the training data (i.e., do a PCA on the covariance matrix of $M$, where $M_{i,j}$ is the $j$-th model's reponse for the $i$-th training example) to produce a set of principal components, $PC = \{PC_1, ..., PC_N\}$.

2. Use a stopping criteria to decide on $k$, the number of principal components to use.

3. Do a least squares regression on the selected components (i.e., include $PC_i$ for $i \leq k$).

4. Derive the weights, $\alpha_i$, for the original learned models by expanding

$$f_{PCR*} = \beta_1 PC_1 + ... + \beta_k PC_k$$

according to

$$PC_i = \gamma_{i,0} f_0 + ... + \gamma_{i,N} f_N,$$

and simplifying for the coefficients of $f_i$. Note that $\gamma_{i,j}$ is the $j$-th coefficient of the $i$-th principal component.

The second step is very important because choosing too few or too many principal components may result in underfitting or overfitting, respectively. Ten-fold cross-validation is used to select $k$ here.

Examining the spectrum of (N) weight sets derived in step four reveals that PCR* provides a continuum of weight sets spanning from highly constrained (i.e., weights generated from $PCR_1$ satisfy all three regularization constraints) to completely unconstrained (i.e., $PCR_N$ is equivalent to unconstrained linear regression). To see that the weights, $\alpha$, derived from $PCR_1$ are (nearly) uniform, recall that the first principal component accounts for where the learned models agree. Because the learned models are all fairly accurate they agree quite often so their first principal component weights, $\gamma_{1,*}$ will be similar. The $\gamma$-weights are in turn multiplied by a constant when $PCR_1$ is regressed upon. Thus, the resulting $\alpha_i$'s will be fairly uniform. The later principal components serve as refinements to those already included producing less constrained weight sets until finally $PCR_N$ is included resulting in an unconstrained estimator much like LR, LRC and GEM.

## 4   EXPERIMENTAL RESULTS

The set of learned models, $\mathcal{F}$, were generated using Backpropogation [Rumelhart, 1986]. For each dataset, a network topology was developed which gave good performance. The collection of networks built differed only in their initial weights[5].

Three data sets were chosen: *cpu* and *housing* (from the UCI repository), and *bodyfat* (from the Statistics Library at Carnegie Mellon University). Due to space limitation, the data sets reported on were chosen because they were representative of the basic trends found in a larger collection of datasets. The combining methods evaluated consist of all the methods discussed in Sections 2 and 3, as well as $PCR_1$ and $PCR_N$ (to demonstrate PCR*'s most and least regularized weight sets,

Table 1: Results

| Data | bodyfat | | cpu | | housing | |
|---|---|---|---|---|---|---|
| N | 10 | 50 | 10 | 50 | 10 | 50 |
| BEM | 1.03 | 1.04 | **38.57** | **38.62** | 2.79 | 2.77 |
| GEM | **1.02** | **0.86** | 46.59 | 227.54 | **2.72** | **2.57** |
| LR | **1.02** | 3.09 | 44.9 | 238.0 | **2.72** | 6.44 |
| RIDGE | **1.02** | 0.826 | 44.8 | 191.0 | **2.72** | **2.55** |
| GD | **1.03** | 1.04 | 38.9 | 38.8 | 2.79 | 2.77 |
| EGPM | 1.03 | 1.07 | **38.4** | **38.0** | 2.77 | 2.75 |
| PCR$_1$ | 1.04 | 1.05 | 39.0 | 39.0 | 2.78 | 2.76 |
| PCR$_N$ | **1.02** | 0.848 | 44.8 | 249.9 | **2.72** | **2.57** |
| PCR* | **0.99** | **0.786** | 40.3 | 40.8 | **2.70** | **2.56** |

respectively). The more computationally intense procedures based on stacking and bootstrapping proposed by [Leblanc and Tibshirani, 1993, Breiman, 1992] were not evaluated here because they required many more models (i.e., neural networks) to be generated for each of the elements of $\mathcal{F}$.

There were 20 trials run for each of the datasets. On each trial the data was randomly divided into 70% training data and 30% test data. These trials were rerun for varying sizes of $\mathcal{F}$ (i.e., 10 and 50, respectively). As more models are included the linear dependence amongst them goes up showing how well the multicollinearity problem is handled[6]. Table 1 shows the average residual errors for the each of the methods on the three data sets. Each row is a particular method and each column is the size of $\mathcal{F}$ for a given data set. Bold-faced entries indicate methods which were *not* significantly different from the method with the lowest error (via two-tailed paired t-tests with $p \leq 0.05$).

PCR* is the only approach which is among the leaders for all three data sets. For the *bodyfat* and *housing* data sets the weights produced by BEM, PCR$_1$, GD, and EG+- tended to be too constrained, while the weights for LR tended to be too unconstrained for the larger collection of models. The less constrained weights of GEM, LR, RIDGE, and PCR$_N$ severely harmed performance in the *cpu* domain where uniform weighting performed better.

The biggest demonstration of PCR*'s robustness is its ability to gravitate towards the more constrained weights produced by the earlier principal components when appropriate (i.e., in the *cpu* dataset). Similarly, it uses the less constrained principal components closer to PCR$_n$ when it is preferable as in the *bodyfat* and *housing* domains.

## 5   CONCLUSION

This investigation suggests that the principal components of a set of learned models can be useful when combining the models to form an improved estimator. It was demonstrated that the principal components provide a continuum of weight sets from highly regularized to unconstrained. An algorithm, PCR*, was developed which attempts to automatically select the subset of these components which provides the lowest prediction error. Experiments on a collection of domains demonstrated PCR*'s ability to robustly handle redundancy in the set of learned models. Future work will be to improve upon PCR* and expand it to the classification task.

## Footnotes

[1]A learned model may be anything from a decision/regression tree to a neural network.

[2]Optimal here refers to weights which minimize mean square error for the training data.

[3]Actually, it is a form of linear regression without the intercept term. The more general form, denote by LRC, would be formulated the same way but with member, $f_0$ which always predicts 1. According to [Leblanc and Tibshirani, 1993] having the extra constant term will not be necessary (i.e., it will equal zero) because in practice, $E[f_i(x)] = E[f(x)]$.

[4]Note that the constraint, $\sum_{i=1}^{N} \alpha_i = 1$, for GEM is a form of *regularization* [Leblanc and Tibshirani, 1993]. The purpose of regularizing the weights is to provide an estimate which is less biased by the training sample. Thus, one would not expect GEM and LR to produce identical weights.

[5]There was no extreme effort to produce networks with more decorrelated errors. Even with such networks, the issue of extreme multicollinearity would still exist because $E[f_i(x)] = E[f_j(x)]$ for all $i$ and $j$.

[6]This is verified by observing the eigenvalues of the principal components and values in the covariance matrix of the models in $\mathcal{F}$

# References

[Breiman *et al*, 1984] Breiman, L., Friedman, J.H., Olshen, R.A. & Stone, C.J. (1984). *Classification and Regression Trees.* Belmont, CA: Wadsworth.

[Breiman, 1992] Breiman, L. (1992). Stacked Regression. Dept of Statistics, Berkeley, TR No. 367.

[Chan and Stolfo, 1995] Chan, P.K., Stolfo, S.J. (1995). A Comparative Evaluation of Voting and Meta-Learning on Partitioned Data *Proceedings of the Twelvth International Machine Learning Conference* (90-98). San Mateo, CA: Morgan Kaufmann.

[Draper and Smith, 1981] Draper, N.R., Smith, H. (1981). *Applied Regression Analysis.* New York, NY: John Wiley and Sons.

[Kivinen and Warmuth, 1994] Kivinen, J., and Warmuth, M. (1994). Exponentiated Gradient Descent Versus Gradient Descent for Linear Predictors. Dept. of Computer Science, UC-Santa Cruz, TR No. ucsc-crl-94-16.

[Krogh and Vedelsby, 1995] Krogh, A., and Vedelsby, J. (1995). Neural Network Ensembles, Cross Validation, and Active Learning. In *Advances in Neural Information Processing Systems 7.* San Mateo, CA: Morgan Kaufmann.

[Hansen and Salamon, 1990] Hansen, L.K., and Salamon, P. (1990). Neural Network Ensembles. *IEEE Transactions on Pattern Analysis and Machine Intelligence, 12* (993-1001).

[Leblanc and Tibshirani, 1993] Leblanc, M., Tibshirani, R. (1993) Combining estimates in regression and classification Dept. of Statistics, University of Toronto, TR.

[Meir, 1995] Meir, R. (1995). Bias, variance and the combination of estimators. In *Advances in Neural Information Processing Systems 7.* San Mateo, CA: Morgan Kaufmann.

[Merz, 1995] Merz, C.J. (1995) Dynamical Selection of Learning Algorithms. In Fisher, C. and Lenz, H. (Eds.) Learning from Data: Artificial Intelligence and Statistics, 5). Springer Verlag

[Montgomery and Friedman 1993] Mongomery, D.C., and Friedman, D.J. (1993). Prediction Using Regression Models with Multicollinear Predictor Variables. IIE Transactions, vol. 25, no. 3 73–85.

[Opitz and Shavlik, 1995] Opitz, D.W., Shavlik, J.W. (1996). Generating Accurate and Diverse Members of a Neural-Network Ensemble. Advances in Neural and Information Processing Systems 8. Touretzky, D.S., Mozer, M.C., and Hasselmo, M.E., eds. Cambridge MA: MIT Press.

[Perrone and Cooper, 1992] Perrone, M. P., Cooper, L. N., (1993). When Networks Disagree: Ensemble Methods for Hybrid Neural Networks. *Neural Networks for Speech and Image Processing*, edited by Mammone, R. J.. New York: Chapman and Hall.

[Rumelhart, 1986] Rumelhart, D. E., Hinton, G. E., & Williams, R. J. (1986). Learning Interior Representation by Error Propagation. *Parallel Distributed Processing, 1* 318–362. Cambridge, MASS.: MIT Press.

[Tresp, 1995] Tresp, V., Taniguchi, M. (1995). Combining Estimators Using Non-Constant Weighting Functions. In *Advances in Neural Information Processing Systems 7.* San Mateo, CA: Morgan Kaufmann.

[Wolpert, 1992] Wolpert, D. H. (1992). Stacked Generalization. *Neural Networks, 5,* 241–259.
